# A Log-Domain Implementation of the Diffusion Network in Very Large Scale Integration

**Yi-Da Wu, Shi-Jie Lin, and Hsin Chen**
Department of Electrical Engineering
National Tsing Hua University
Hsinchu, Taiwan 30013
{ydwu;hchen}@ee.nthu.edu.tw

## Abstract

The Diffusion Network(DN) is a stochastic recurrent network which has been shown capable of modeling the distributions of continuous-valued, continuous-time paths. However, the dynamics of the DN are governed by stochastic differential equations, making the DN unfavourable for simulation in a digital computer. This paper presents the implementation of the DN in analogue Very Large Scale Integration, enabling the DN to be simulated in real time. Moreover, the log-domain representation is applied to the DN, allowing the supply voltage and thus the power consumption to be reduced without limiting the dynamic ranges for diffusion processes. A VLSI chip containing a DN with two stochastic units has been designed and fabricated. The design of component circuits will be described, so will the simulation of the full system be presented. The simulation results demonstrate that the DN in VLSI is able to regenerate various types of continuous paths in real-time.

## 1  Introduction

In many implantable biomedical microsystems [1, 2], an embedded system capable of recognising high-dimensional, time-varying signals have been demanded. For example, recognising multi-channel neural activity on-line is important for implantable brain-machine interfaces to avoid transmitting all data wirelessly, or to control prosthetic devices and to deliver bio-feedbacks in real-time [3].

The Diffusion Network (DN) proposed by Movellan is a stochastic recurrent network whose stochastic dynamics can be trained to model the probability distributions of continuous-time paths by the Monte-Carlo Expectation-Maximisation (EM) algorithm [4, 5]. As stochasticity is useful for generalising the natural variability in data [6, 7], the DN is further shown suitable for recognising noisy, continuous-time biomedical data [8]. However, the stochastic dynamics of the DN is defined by a set of continuous-time, stochastic differential equations (SDEs). The speed of simulating stochastic differential equations in a digital computer is inherently limited by the serial processing and numerical iterations of the computer. Translating the DN into analogue circuits is thus of great interests for simulating the DN in real time by exploiting the natural, differential current-voltage (I-V) relationship of capacitors [9].

This paper presents the implementation of the DN in analogue Very Large Scale Integration (VLSI). To minimise the power consumption, the power supply voltage is only 1.5V, and most transistors are operated in subthreshold regions. As the reduced supply voltage limits directly the dynamic range available for voltages across capacitors, the log-domain representation proposed in [10] is applied to the DN, allowing diffusion processes to be simulated in a limited voltage ranges. After a brief

introduction to the DN, the following sections will derive the log-domain representation of the DN and describe its corresponding implementation in analogue VLSI.

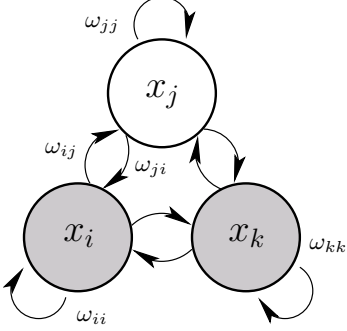

Figure 1: The architecture of a Diffusion Network with one visible and two hidden units

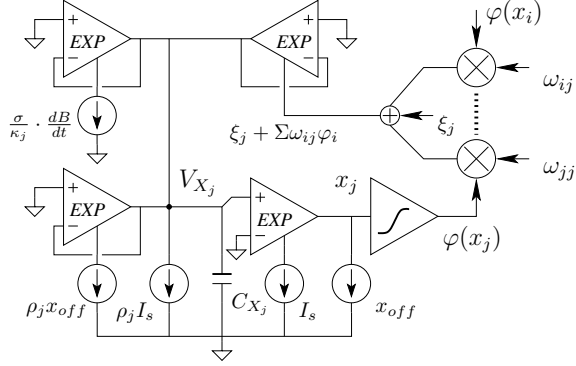

Figure 2: The block diagram of a DN unit in VLSI

## 2 The Diffusion Network

As shown in Fig. 1, the DN comprises $n$ continuous-time, continuous-valued stochastic units with fully recurrent connections. The state of the $j^{th}$ unit at time $t$, $x_j(t)$, is governed by

$$\frac{dx_j(t)}{dt} = \mu_j\big(x_j(t)\big) + \sigma \cdot \frac{dB(t)}{dt} \tag{1}$$

where $\mu_j(t)$ is a deterministic *drift* term given in (2), $\sigma$ a constant, and $dB(t)$ the *Brownian* motion. The Brownian motion introduces the stochasticity, enriching greatly the representational capability of the DN [5].

$$\mu_j\big(x_j(t)\big) = \kappa_j \cdot \left[ -\rho_j x_j(t) + \xi_j + \sum_{i=1}^{n} \omega_{ij} \cdot \varphi\big(x_i(t)\big) \right] \tag{2}$$

$\omega_{ij}$ defines the *connection weight* from unit $i$ to unit $j$. $\kappa_j^{-1}$ and $\rho_j^{-1}$ represent the *input capacitance* and *transmembrane resistance*, respectively, of the $j^{th}$ unit. $\xi_j$ is the *input bias*, and $\varphi$ is the *sigmoid* function given as

$$\varphi(x_j; a) = -1 + \frac{2}{1 + e^{-ax_j}} = \tanh\left(\frac{a}{2} x_j\right) \tag{3}$$

where $a$ adapts the slope of the sigmoid function. As shown in Fig. 1, the DN contains both visible(white) and hidden(grey) stochastic units. The learning of the DN aims to regenerate at visible units the probability distribution of a specific set of continuous paths. The number of visible units thus equals the dimension of the data to be modeled, while the minimum number of hidden units required for modeling data satisfactorily is identified by experimental trials. During training, visible units are "clamped" to the dynamics of the training dataset, and the dynamics of hidden units are Monte-Carlo sampled for estimating optimal parameters $(\omega_{ij}, \kappa_j, \rho_j, \xi_j)$ that maximise the expectation of training data [5]. After training, all units are given initial values at $t = 0$ only to sample the dynamics modeled by the DN. The similarity between the dynamics of visible units and those of training data indicate how well the DN models the data.

### 2.1 Log-domain translation

To maximise the dynamic ranges for diffusion processes in VLSI, the stochastic state $x_j(t)$ is represented as a current and then logarithmically-compressed into a voltage $V_{Xj}$ in VLSI [11]. The logarithmic compression allows $x_j(t)$ to change over three decades within a limited voltage range for $V_{Xj}$. The voltage representation $V_{Xj}$ further facilitates the exploitation of the nature, differential (I-V) relationship of a capacitor to simulate SDEs in real-time and in parallel.

The logarithmic relationship between $x_j(t)$ and $V_{Xj}$ can be realised by the exponential I-V characteristics of a MOS transistor in subthreshold operation [12]. To keep $x_j(t)$ a non-negative value (current) in VLSI, an offset $x_{off}$ is added to $x_j(t)$, resulting in the following relationship between $x_j(t)$ and $V_{Xj}$.

$$x_j + x_{off} \equiv I_S \cdot e^{\alpha V_{Xj}}, \ dx_j = \alpha I_S \cdot e^{\alpha V_{Xj}} \cdot dV_{Xj} \tag{4}$$

where $I_s$ and $\alpha$ are process-dependent constants extractable from simulated I-V curves of transistors. Substituting Eq. (4) into Eq. (1) then translates the diffusion process in Eq. (1) into the following equation.

$$C_{Xj} \cdot \frac{dV_{Xj}}{dt} = \left[ \xi_j + \sum_{i=1}^{n} \omega_{ij} \varphi(x_i) \right] \cdot e^{-\alpha V_{Xj}} + \frac{\sigma}{\kappa_j} \frac{dB_j(t)}{dt} \cdot e^{-\alpha V_{Xj}} + \rho_j x_{off} \cdot e^{-\alpha V_{Xj}} - \rho_j I_S \tag{5}$$

where $C_{Xj}$ equals $\alpha/\kappa_j$. Fig. 2 illustrates the block diagram for implementing Eq. (5) in VLSI. $C_{Xj}$ is a capacitor and $V_{Xj}$ the voltage across the capacitor. Each term on the right hand side of Eq. (5) then corresponds to a current flowing into $C_{Xj}$. Let $(V_P - V_N)$ and $I_{VAR}$ represent the differential input voltage and the input current of an EXP-element, respectively. Each EXP-element in Fig. 2 produces an output current of $I_{out} = I_{VAR} \cdot e^{\alpha(V_P - V_N)}$. Therefore, the EXP-elements implement the first three terms multiplied with $e^{-\alpha V_{Xj}}$ in accordance with Eq. (5). The last term, $\rho_j I_S$, is a constant and is thus implemented by a constant current source. Finally, the sigmoid circuit transforms $x_j$ into $\varphi(x_j)$ and the multipliers output a total current proportional to $\sum_{i=1}^{n} \omega_{ij} \cdot \varphi(x_i)$.

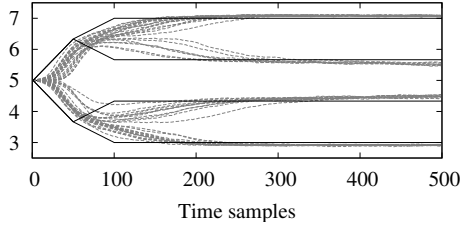

Figure 3: The stochastic dynamics (gray lines) regenerated by the DN trained on the bifurcating curves (black lines).

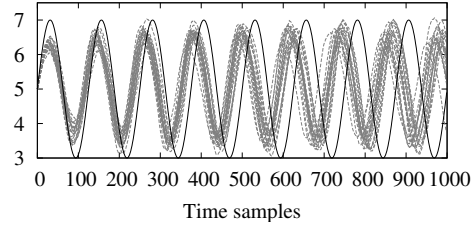

Figure 4: The stochastic dynamics (gray lines) regenerated by the DN trained on the sinusoidal curve (the black line).

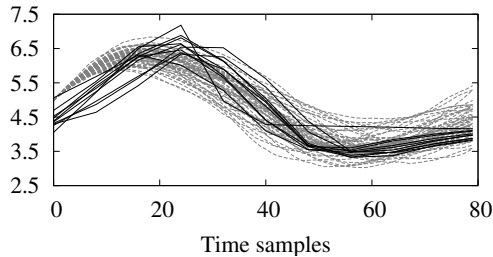

Figure 5: The stochastic dynamics (gray lines) regenerated by the DN trained on the QRS segments of electrocardiograms (black lines).

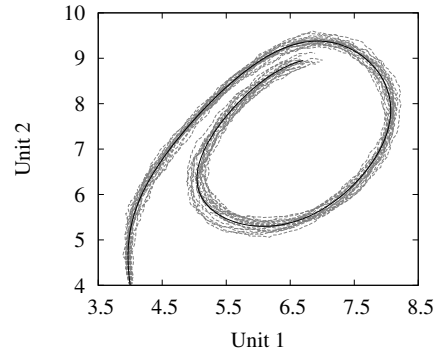

Figure 6: The stochastic dynamics (gray lines) regenerated by the DN trained on the handwritten $\rho$ (the black line).

## 2.2 Adapting $\rho_j$ instead of $\kappa_j$

The DN has been shown capable of modeling various distributions of continuous paths by adapting $w_{ij}, \xi_j$, and $\kappa_j$ in [5]. An adaptable $\kappa_j$ corresponds to an adaptable $C_{Xj}$, but a tunable capacitor with a wide linear range is not easy to implement in VLSI. As Eq. (2) indicates that $\rho_j$ is complementary

to $\kappa_j$ in determining the "time constant" of the dynamics of the unit $j$, the possibility of adapting $\rho_j$ instead of $\kappa_j$ is investigated by Matlab simulation.

With $\kappa_j = 1$, the DN was trained to model different data by adapting $\omega_{ij}$, $\xi_j$, and $\rho_j$ for 100 epochs. A DN with one visible and one hidden units was proved capable of regenerating the dynamics of bifurcating curves (Fig. 3), sinusoidal waves (Fig. 4), and electrocardiograms (Fig. 5). Moreover, a DN with only two visible units was able to regenerate the handwritten $\rho$ satisfactorily, as illustrated in Fig. 6. The promising results supported the suggestion that adapting $\rho_j$ instead of $\kappa_j$ also allowed the DN to model different data. As a variable $\rho_j$ simply corresponded to a tunable current source $\rho_j I_S$ in Fig. 2, the VLSI implementation was greatly simplified.

## 2.3   Parameter mappings

Table 1 summarises the parameter mappings between the numerical simulation and the VLSI implementation. All variables except for $V_{Xj}$ in Fig. 2 are represented as currents in VLSI. The unit currents ($I_{unit}$) of $x_j$, $\omega_{ij}$, and $\xi_j$ are defined as 10 nA to match the current scales of transistors in subthreshold operation, as well as to reduce the power consumption. Moreover, extensive simulations indicate that the dynamic ranges required for modeling various data are $[-3, 5]$ for $x_j$ and $[-30, 30]$ for $\omega_{ij}$. With $x_{off} = 5$ in Eq. (4), i.e. $x_{off} = 50nA$ in VLSI, $V_{Xj}$ ranges from 773 to 827 mV. While the diffusion process in Eq. (1) is iterated with $\Delta t = 0.05$ in numerical simulation, $\Delta t = 0.05$ is set to be 5 $\mu$s in VLSI, corresponding to a reasonable sampling rate (200kHz) at which most instruments can sample multiple channels(units) simultaneously. Finally, the unit capacitance for $1/\kappa_j$ is calculated as $C_{unit} = I_{unit} \cdot \Delta t_{unit} / V_{Xj,unit}$, equaling 1 pF and resulting in $C_{Xj} = \alpha \cdot C_{unit} = 30$ pF.

Table 1: Parameter mappings between numerical simulation and VLSI implementation

| parameter | numeric | circuit | comment |
|---|---|---|---|
| $x_j$ | -3~5 | -30~50 nA | $I_{unit} = 10$ nA |
| $x_{off}$ | 5 | 50 nA | offset term in Eq. (4) |
| $V_{Xj}$ | 0.773~0.827 | 773~827 mV | $V_{Xj,unit} = 1$ V |
| $\omega, \xi$ | -30~30 | -300~300 nA | $I_{unit} = 10$ nA |
| $\varphi(x_j)$ | -1~1 | -400~400 nA | activation function |
| $C_{Xj}$ | $\alpha/\kappa_j = 30$ | 30 pF | $C_{unit} = 1$ pF |
| $\Delta$t | 0.05 | 5 $\mu$s | $t_{unit} = 0.1$ ms |
| $\rho$ | 0.5~2 | 0.5~2 | |

## 3   Circuit implementation

A DN with two stochastic units have been designed with the CMOS 0.18 $\mu$m technology provided by the Taiwan Semiconductor Manufacturing Company (TSMC). The following subsections introduce the design of each component circuit.

### 3.1   The EXP element

Fig. 7(b) shows the schematics of the EXP element. With M1 and M2 operated in the subthreshold region, the output current is given as

$$I_{out} = I_B \cdot \exp\left(\frac{1}{nU_T}(V_P - V_N)\right) \tag{6}$$

where $U_T$ denotes the thermal voltage and $n$ the *subthreshold slope factor*. Comparing Eq. (6) with Eq. (4) reveals that $\alpha = 1/nU_T$. As the drain current ($I_d$) of a transistor in subthreshold operation is exponentially proportional to its gate-to-source voltage ($V_{GS}$) as $I_d \propto e^{V_{GS}/nU_T}$, $\alpha = 1/nU_T$ is extracted to be 30 by plotting $\log(I_d)$ versus $V_{GS}$ in SPICE.

Transistors M3-M5 form an active biasing circuit that sinks $I_B + I_{out}$. By adjusting the gate voltage of M3 through the negative feedback, $I_{out}$ is allowed to change over several decades. In addition,

$n$ actually depends on the gate voltage and introduces variability to $\alpha$ [13]. To prevent the variable $\alpha$ from introducing simulation errors, all EXP elements of the DN unit are biased with a constant $I_B = 100$ nA. As shown by Fig. 7(a), $I_{out}$ of each element is then re-scaled by the one-quadrant current multiplier basing on translinear loops (Fig. 7(c)) [13] to produce $I'_{out} = I_{out} \times I_{VAR}/I_B$, where $I_{VAR}$ represents the current input to each element in Fig. 2 (e.g.$\Sigma\omega\varphi$ or $\rho x_{off}$).

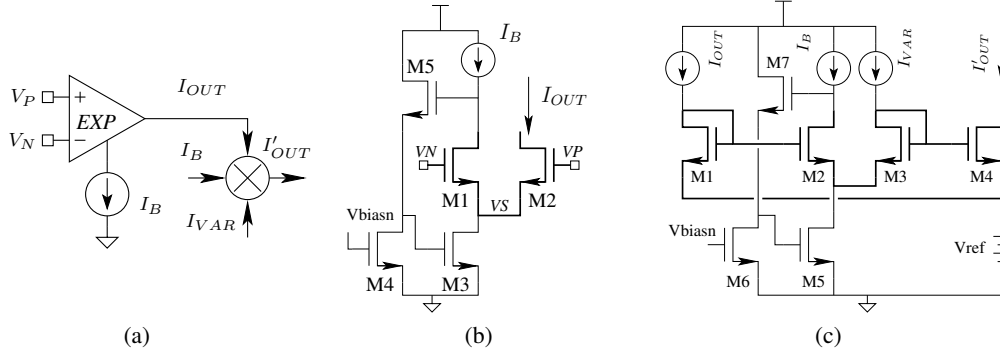

Figure 7: The circuit diagram of the EXP element.

## 3.2  Current multipliers

Four-quadrant multipliers basing on translinear loops [13] are employed to calculate $\Sigma\omega_{ij}\varphi(x_i)$ in Eq. (5). Both $\omega_{ij}$ and $\varphi(x_i)$ are represented by differential currents as

$$\omega_{ij} = I_{\omega_+} - I_{\omega_-}, \;\; \varphi(x_i) = I_{\varphi_+} - I_{\varphi_-} \tag{7}$$

Let the differential current $(I_{Z+} - I_{Z-})$ represents the multiplier's output and $I_U$ represent a unit current. Eq. (8) indicates that the four-quadrant multiplication can be composed of four one-quadrant multipliers in Fig. 7(c), as illustrated in Fig. 8.

$$I_{Z+} \cdot I_U - I_{Z-} \cdot I_U = (I_{\omega_+} \cdot I_{\varphi_+} + I_{\omega_-} \cdot I_{\varphi_-}) - (I_{\omega_+} \cdot I_{\varphi_-} + I_{\omega_-} \cdot I_{\varphi_+}) \tag{8}$$

Fig. 9 shows the simulation result of the four-quadrant multiplier, exhibiting satisfactory linearity over the dynamic ranges required in Table 1.

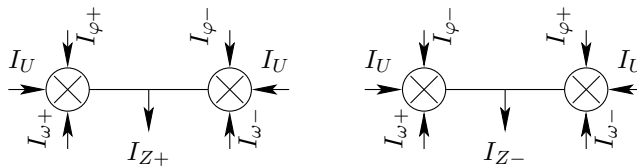

Figure 8: The four-quadrant current multiplier

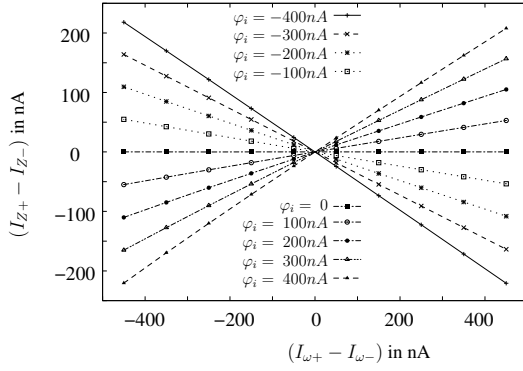

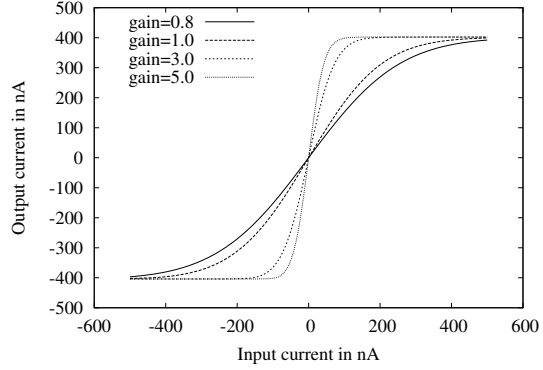

Figure 9: The simulation results of the four-quadrant current multiplier

Figure 10: The simulation result of the sigmoid circuit with different $V_a$

### 3.3 Sigmoid function $\varphi(\cdot)$

Fig. 11 shows the block diagram for implementing the sigmoid function in Eq. (3). The current $I_{Xi}$ representing $x_i$ is firstly converted into a voltage $V_i$ by the the operational amplifier(OPA) with a voltage-controlled active resistor (VCR) proposed in [14]. $V_i$ is then sent to an operational transconductance amplifier(OTA) in subthreshold operation, producing an output current of

$$I_s = I_B \tanh\Big(\frac{1}{2nU_T}(V_i - V_{ref})\Big) \tag{9}$$

Since $V_i - V_{ref} = R_i \cdot I_{xi}$, with $R_i$ representing the resistance of the VCR, the voltage $V_a$ adapts $R_i$ and thus the slope of the sigmoid function. Finally, the $2^{nd}$ generation current conveyor (CCII) in Fig. 12 [15] converts the current $I_s$ into a pair of differential currents ($I_{OUTN}$, $I_{OUTP}$) ranging between $-400$ nA and $+400$ nA. The differential currents are then duplicated for the inputs of four-quadrant multipliers of all DN units.

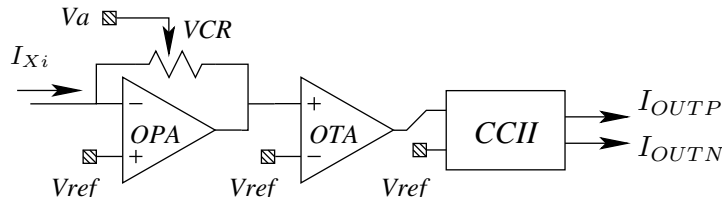

Figure 11: The block diagram of the sigmoid circuit.

### 3.4 Capacitor amplification

As $C_{Xi} = 30$ pF requires considerable chip area, $C_{Xi}$ is implemented by the circuit in Fig. 13, utilising the Miller effect to amplify the capacitance. Let $A$ denote the gain of the amplifier. The effective capacitance between X and Y is $(1 + A) \cdot C_X$. Fig. 13 also shows the schematics of the amplifier whose gain is designed to be 2. As a result, $C_X = 10$ pF is sufficient for providing an effective $C_{Xi}$ of 30 pF.

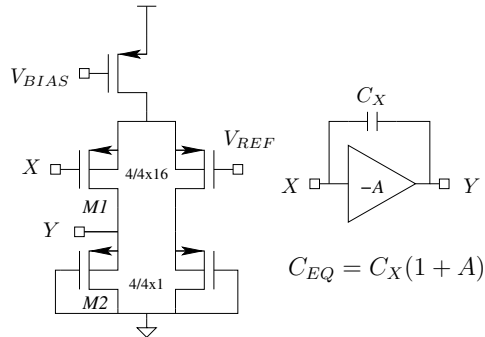

Figure 13: The circuit diagram of the capacitor amplified by the Miller effect.

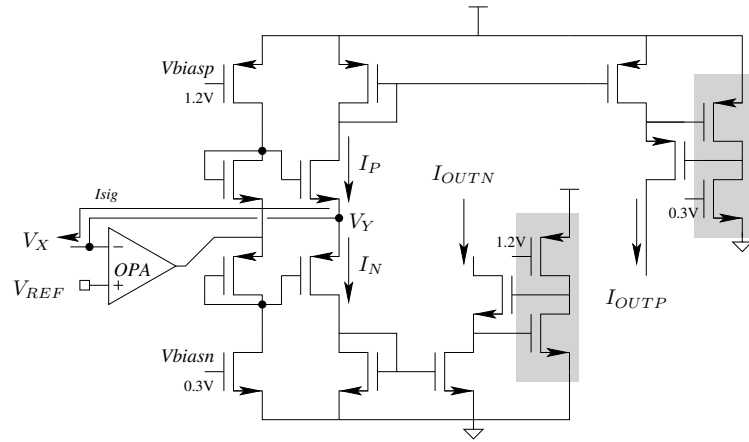

Figure 12: The circuit diagram of the single-to-differential current conveyor

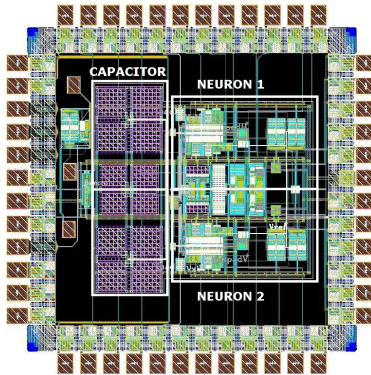

| Technology | 1P6M 0.18 $\mu$m CMOS |
|---|---|
| Power Supply | 1.5 Volts |
| Power Consumption | 345 $\mu$Watts |
| Num. of Units | 2 |
| Chip Area | $1.368 \times 1.368$mm$^2$ (including pads) |
| Capability | 1D/2D continuous paths |
| Max. Bandwidth | 1.6 kHz |

Figure 14: The chip layout and its specification.

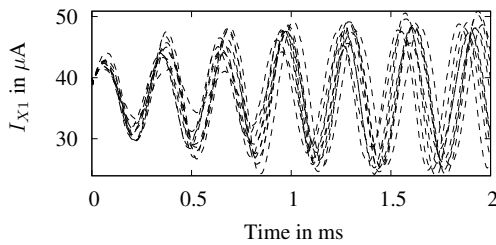

Figure 15: The sinusoidal dynamics regenerated by the DN chip in post-layout simulation (10 trials).

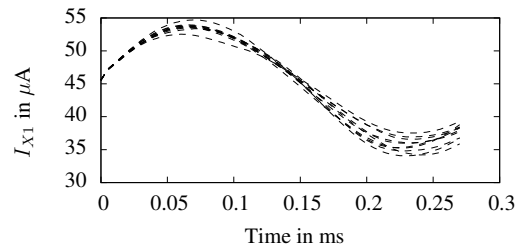

Figure 16: The electrocardiogram dynamics regenerated by the DN chip in post-layout simulation (10 trials).

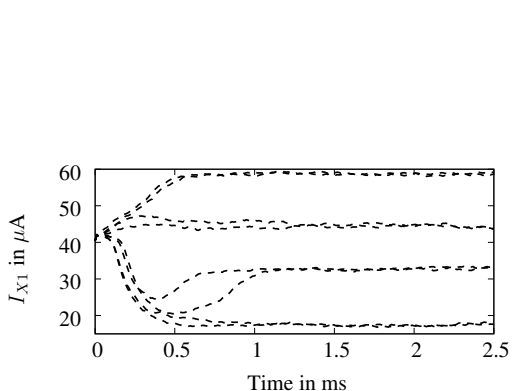

Figure 17: The bifurcating dynamics regenerated by the DN chip in post-layout simulation (8 trials).

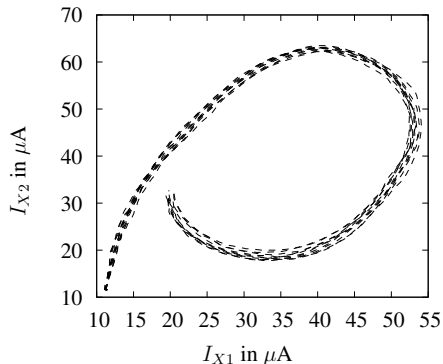

Figure 18: The handwritten $\rho$ regenerated by the DN chip in post-layout simulation (10 trials).

## 4 The Diffusion Network in VLSI

Fig. 14 shows the chip layout of the log-domain implementation of the DN with two stochastic units, so is the specification shown. The area of the core circuit and the capacitors are 0.306 mm$^2$ and 0.384 mm$^2$, respectively. The total power consumption is merely 345 $\mu$W, by the merit of low supply voltage (1.5V) and subthreshold operation. The chip has been taped out for fabrication with the CMOS 0.18 $\mu$m Technology by the TSMC. The post-layout simulations are shown in Fig. 15$-$18 and described as follows.

With one unit functioning as a visible unit and the other as a hidden unit, the parameters of the DN was programmed to regenerate the one-dimensional paths in Sec. 2.2. The noise current $\frac{\sigma}{\kappa} \cdot \frac{dB}{dt}$ was simulated by a piecewise-linear current source with random amplitudes in the SPICE. As shown by Fig. 15-17, the visible unit was capable of regenerating the sinusoidal waves, the electrocardio-grams, and the bifurcating curves with negligible differences from Fig. 3-5. Moreover, as both units functioned as visible units, the DN was capable of regenerating the handwritten $\rho$ as Fig. 18. These promising results demonstrate the capability of the DN chip to model the distributions of different continuous paths reliably and power-efficiently. After chip is fabricated in August, the chip will be tested and the measurement results will be presented in the conference.

## 5 Conclusion

The log-domain representation of the Diffusion Network has been derived and translated into ana-logue VLSI circuits. Based on well-defined parameter mappings, the DN chip is proved capable of regenerating various types of continuous paths, and the log-domain representation allows the dif-fusion processes to be simulated in real-time and within a limited dynamic range. In other words, analogue VLSI circuits are proved useful for solving (simulating) multiple SDEs in real-time and in a power-efficient manner. After verifying the chip functionality, a DN chip with a scalable number of units will be further developed for recognising multi-channel, time-varying biomedical signals in implantable microsystems.

**Acknowledgments**

The authors thank National Chip Implementation Center (CIC) for fabrication services, and Mr. C.-M. Lai and S.-C. Sun for helpful discussions.

# References

[1] G. Iddan, G. Meron, A. Glukhovsky, and P. Swain, "Wireless capsule endoscopy," *Nature*, vol. 405, no. 6785, p. 417, July 2000.

[2] T. W. Berger, M. Baudry, J.-S. L. Roberta Diaz Brinton, V. Z. Marmarelis, A. Y. Park, B. J. Sheu, and A. R. Tanguay, JR., "Brain-implantable biomimetic electronics as the next era in neural prosthetics," *Proc. IEEE*, vol. 89, no. 7, pp. 993–1012, July 2001.

[3] M. A. Lebedev and M. A. L. Nicolelis, "Brain-machine interfaces: past, present and future," *Trends in Neuroscience*, vol. 29, no. 9, pp. 536–546, 2006.

[4] J. R. Movellan, "A learning theorem for networks at detailed stochastic equilibrium," *Neural Computation*, vol. 10, pp. 1157–1178, July 1998.

[5] J. R. Movellan, P. Mineiro, and R. J.Williams, "A Monte Carlo EM approach for partially observable diffusion processes: Theory and applications to neural networks," *Neural Computation*, vol. 14, pp. 1507–1544, July 2002.

[6] H. Chen and A. F. Murray, "A continuous restricted Boltzmann machine with an implementable training algorithm," *IEE Proc. of Vision, Image and Signal Processing*, vol. 150, no. 3, pp. 153–158, 2003.

[7] D. F. Specht, "Probabilistic neural networks," *Neural Networks*, vol. 3, no. 1, pp. 109–118, 1990.

[8] Y. S. Hsu, T. J. Chiu, and H. Chen, "Real-time recognition of continuous-time biomedical signals using the diffusion network," in *Proc. of the Int. Joint Conf. on Neural Networks (IJCNN)*, 2008, pp. 2628–2633.

[9] L. O. Chua, T. Roska, T. Kozek, and A. Zarandy, "CNN universal chips crank up the computing power," *IEEE Circuits and Devices Mag.*, vol. 12, no. 4, pp. 18–28, July 1996.

[10] T. Serrano-Gotarredona and B. Linares-Barranco, "Log-domain implementation of complex dynamics reaction-diffusion neural networks," *IEEE Trans. Neural Networks*, vol. 14, pp. 1337–1355, Sept. 2003.

[11] D. R. Frey, "Exponential state space filters: A generic current mode design strategy," *IEEE Trans. Circuits Syst. I*, vol. 43, pp. 34–42, Jan. 1996.

[12] E. Vittoz and J. Fellrath, "CMOS analog integrated circuits based on weak inversion operation," *IEEE J. Solid-State Circuits*, vol. 12, pp. 224–231, June 1977.

[13] S.-C. Liu, J. Kramer, G. Indiveri, T. Delbrück, and R. Douglas, *Analog VLSI: Circuits and Principles*. The MIT Press, 2002.

[14] M. Banu and Y. Tsividis, "Floating voltage-controlled resistors in CMOS technology," *Electronics Letters*, vol. 18, no. 15, pp. 678–679, July 1982.

[15] C. Toumazou, F. J. Lidgey, and D. G. Haigh, *Analogue IC Design: The Current-Mode Approach*. Peter Peregrinus Ltd, 1990.

